# Linear Operator for Object Recognition

**Ronen Basri**         **Shimon Ullman***
M.I.T. Artificial Intelligence Laboratory
and Department of Brain and Cognitive Science
545 Technology Square
Cambridge, MA 02139

## Abstract

Visual object recognition involves the identification of images of 3-D objects seen from arbitrary viewpoints. We suggest an approach to object recognition in which a view is represented as a collection of points given by their location in the image. An object is modeled by a set of 2-D views together with the correspondence between the views. We show that any novel view of the object can be expressed as a linear combination of the stored views. Consequently, we build a linear operator that distinguishes between views of a specific object and views of other objects. This operator can be implemented using neural network architectures with relatively simple structures.

## 1 Introduction

Visual object recognition involves the identification of images of 3-D objects seen from arbitrary viewpoints. In particular, objects often appear in images from previously unseen viewpoints. In this paper we suggest an approach to object recognition in which rigid objects are recognized from arbitrary viewpoint. The method can be implemented using neural network architectures with relatively simple structures.

In our approach a view is represented as a collection of points given by their location in the image, An object is modeled by a small set of views together with the correspondence between these views. We show that any novel view of the object

can be expressed as a linear combination of the stored views. Consequently, we build a linear operator that distinguishes views of a specific object from views of other objects. This operator can be implemented by a neural network.

The method has several advantages. First, it handles correctly rigid objects, but is not restricted to such objects. Second, there is no need in this scheme to explicitly recover and represent the 3-D structure of objects. Third, the computations involved are often simpler than in previous schemes.

## 2   Previous Approaches

Object recognition involves a comparison of a viewed image against object models stored in memory. Many existing schemes to object recognition accomplish this task by performing a template comparison between the image and each of the models, often after compensating for certain variations due to the different positions and orientations in which the object is observed. Such an approach is called *alignment* (Ullman, 1989), and a similar approach is used in (Fischler & Bolles 1981, Lowe 1985, Faugeras & Hebert 1986, Chien & Aggarwal 1987, Huttenlocher & Ullman 1987, Thompson & Mundy 1987).

The majority of alignment schemes use object-centered representations to model the objects. In these models the 3-D structure of the objects is explicitly represented. The acquisition of models in these schemes therefore requires a separate process to recover the 3-D structure of the objects.

A number of recent studies use 2-D viewer-centered representations for object recognition. Abu-Mostafa & Pslatis (1987), for instance, developed a neural network that continuously collects and stores the observed views of objects. When a new view is observed it is recognized if it is sufficiently similar to one of the previously seen views. The system is very limited in its ability to recognize objects from novel views. It does not use information available from a collection of object views to extend the range of recognizable views beyond the range determined by each of the stored views separately.

In the scheme below we suggest a different kind of viewer-centered representations to model the objects. An object is modeled by a set of its observed images with the correspondence between points in the images. We show that only a small number of images is required to predict the appearance of the object from all possible viewpoints. These predictions are exact for rigid objects, but are not confined to such objects. We also suggest a neural network to implement the scheme.

A similar representation was recently used by Poggio & Edelman (1990) to develop a network that recognizes objects using radial basis functions (RBFs). The approach presented here has several advantages over this approach. First, by using the linear combinations of the stored views rather than applying radial basis functions to them we obtain exact predictions for the novel appearances of objects rather than an approximation. Moreover, a smaller number of views is required in our scheme to predict the appearance of objects from all possible views. For example, when a rigid object that does not introduce self occlusion (such as a wired object) is considered, predicting its appearance from all possible views requires only three views under the LC Scheme and about sixty views under the RBFs Scheme.

## 3   The Linear Combinations (LC) Scheme

In this section we introduce the Linear Combinations (LC) Scheme. Additional details about the scheme can be found in (Ullman & Basri, 1991). Our approach is based on the following observation. For many continuous transformations of interest in recognition, such as 3-D rotation, translation, and scaling, every possible view of a transforming object can be expressed as a linear combination of other views of the object. In other words, the set of possible images of an object undergoing rigid 3-D transformations and scaling is embedded in a linear space, spanned by a small number of 2-D images.

We start by showing that any image of an object undergoing rigid transformations followed by an orthographic projection can be expressed as a linear combination of a small number of views. The coefficients of this combination may differ for the $x$- and $y$-coordinates. That is, the intermediate view of the object may be given by two linear combinations, one for the $x$-coordinates and the other for the $y$-coordinates. In addition, certain functional restrictions may hold among the different coefficients.

We represent an image by two coordinate vectors, one contains the $x$-values of the object's points, and the other contains their $y$-values. In other words, an image $P$ is described by $\mathbf{x} = (x_1, ..., x_n)$ and $\mathbf{y} = (y_1, ..., y_n)$ where every $(x_i, y_i)$, $1 \le i \le n$, is an image point. The order of the points in these vectors is preserved in all the different views of the same object, namely, if $P$ and $P'$ are two views of the same object, then $(x_i, y_i) \in P$ and $(x_i', y_i') \in P'$ are in correspondence (or, in other words, they are the projections of the same object point).

**Claim:**    The set of coordinate vectors of an object obtained from all different viewpoints is embedded in a 4-D linear space.
(A proof is given in Appendix A.)

Following this claim we can represent the entire space of views of an object by a basis that consists of any four linearly independent vectors taken from the space. In particular, we can construct a basis using familiar views of the object. Two images supply four such vectors and therefore are often sufficient to span the space. By considering the linear combinations of the model vectors we can reproduce any possible view of the object.

It is important to note that the set of views of a rigid object does not occupy the entire linear 4-D space. Rather, the coefficients of the linear combinations reproducing valid images follow in addition two quadratic constraints. (See Appendix A.) In order to verify that an object undergoes a rigid transformation (as opposed to a general 3-D affine transformation) the model must consist of at least three snapshots of the object.

Many 3-D rigid objects are bounded with smooth curved surfaces. The contours of such objects change their position on the object whenever the viewing position is changed. The linear combinations scheme can be extended to handle these objects as well. In this cases the scheme gives accurate approximations to the appearance of these objects (Ullman & Basri, 1991).

The linear combination scheme assumes that the same object points are visible in the different views. When the views are sufficiently different, this will no longer hold,

due to self-occlusion. To represent an object from all possible viewing directions (e.g., both "front" and "back"), a number of different models of this type will be required. This notion is similar to the use of different object aspects suggested by Koenderink & Van Doorn (1979). (Other aspects of occlusion are discussed in the next section.)

# 4    Recognizing an Object Using the LC Scheme

In the previous section we have shown that the set of views of a rigid object is embedded in a linear space of a small dimension. In this section we define a linear operator that uses this property to recognize objects. We then show how this operator can be used in the recognition process.

Let $\mathbf{p}_1, ..., \mathbf{p}_k$ be the model views, and $\mathbf{p}$ be a novel view of the same object. According to the previous section there exist coefficients $a_1, ..., a_k$ such that: $\mathbf{p} = \sum_{i=1}^{k} a_i \mathbf{p}_i$. Suppose $L$ is a linear operator such that $L\mathbf{p}_i = \mathbf{q}$ for every $1 \leq i \leq n$ and some constant vector $\mathbf{q}$, then $L$ transforms $\mathbf{p}$ to $\mathbf{q}$ (up to a scale factor), $L\mathbf{p} = (\sum_{i=1}^{k} a_i)\mathbf{q}$. If in addition $L$ transforms vectors outside the space spanned by the model to vectors other then $\mathbf{q}$ then $L$ distinguishes views of the object from views of other objects. The vector $\mathbf{q}$ then serves as a "name" for the object. It can either be the zero vector, in which case $L$ transforms every novel view of the object to zero, or it can be a familiar view of the object, in which case $L$ has an associative property, namely, it takes a novel view of an object and transforms it to a familiar view. A constructive definition of $L$ is given in appendix B.

The core of the recognition process we propose includes a neural network that implements the linear operator defined above. The input to this network is a coordinate vector created from the image, and the output is an indication whether the image is in fact an instance of the modeled object. The operator can be implemented by a simple, one layer, neural network with only feedforward connections, the type presented by Kohonen, Oja, & Lehtiö (1981). It is interesting to note that this operator can be modified to recognize several models in parallel.

To apply this network to the image the image should first be represented by its coordinate vectors. The construction of the coordinate vectors from the image can be implemented using cells with linear response properties, the type of cells encoding eye positions found by Zipser & Andersen (1988). The positions obtained should be ordered according to the correspondence of the image points with the model points. Establishing the correspondence is a difficult task and an obstacle to most existing recognition schemes. The phenomenon of apparent motion (Marr & Ullman 1981) suggests, however, that the human visual system is capable of handling this problem.

In many cases objects seen in the image are partially occluded. Sometimes also some of the points cannot be located reliably. To handle these cases the linear operator should be modified to exclude the missing points. The computation of the updated operator from the original one involves computing a pseudo-inverse. A method to compute the pseudo-inverse of a matrix in real time using neural networks has been suggested by Yeates (1991).

## 5  Summary

We have presented a method for recognizing 3-D objects from 2-D images. In this method, an object-model is represented by the linear combinations of several 2-D views of the object. It has been shown that for objects undergoing rigid transformations the set of possible images of a given object is embedded in a linear space spanned by a small number of views. Rigid transformations can be distinguished from more general linear transformations of the object by testing certain constraints placed upon the coefficients of the linear combinations. The method applies to objects with sharp as well as smooth boundaries.

We have proposed a linear operator to map the different views of the same object into a common representation, and we have presented a simple neural network that implements this operator. In addition, we have suggested a scheme to handle occlusions and unreliable measurements. One difficulty in this scheme is that it requires to find the correspondence between the image and the model views. This problem is left for future research.

The linear combination scheme described above was implemented and applied to a number of objects. Figures 1 and 2 show the application of the linear combinations method to artificially created and real life objects. The figures show a number of object models, their linear combinations, and the agreement between these linear combinations and actual images of the objects. Figure 3 shows the results of applying a linear operator with associative properties to artificial objects. It can be seen that whenever the operator is fed with a novel view of the object for which it was designed it returns a familiar view of the object.

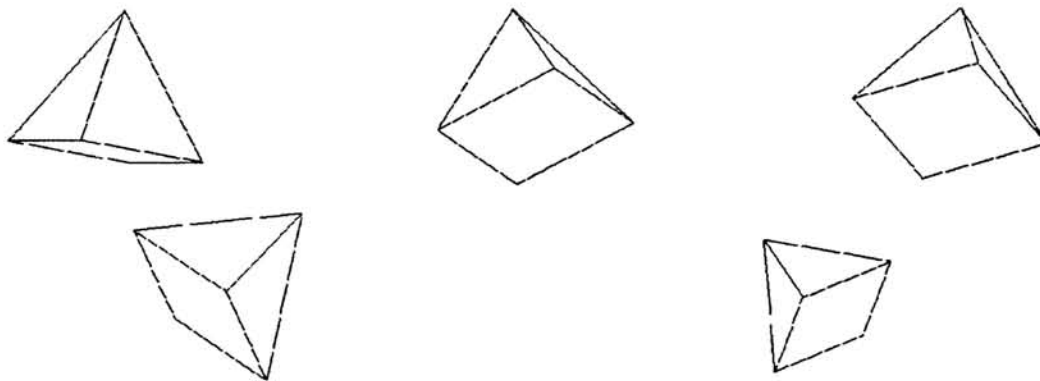

Figure 1: Top: three model pictures of a pyramid. Bottom: two of their linear combinations.

## Appendix A

In this appendix we prove that the coordinate vectors of images of a rigid object lie in a 4-D linear space. We also show that the coefficients of the linear combinations that produce valid images of the object follow in addition two quadratic constraints.

Let $O$ be a set of object points, and let $\mathbf{x} = (x_1, ..., x_n)$, $\mathbf{y} = (y_1, ..., y_n)$, and

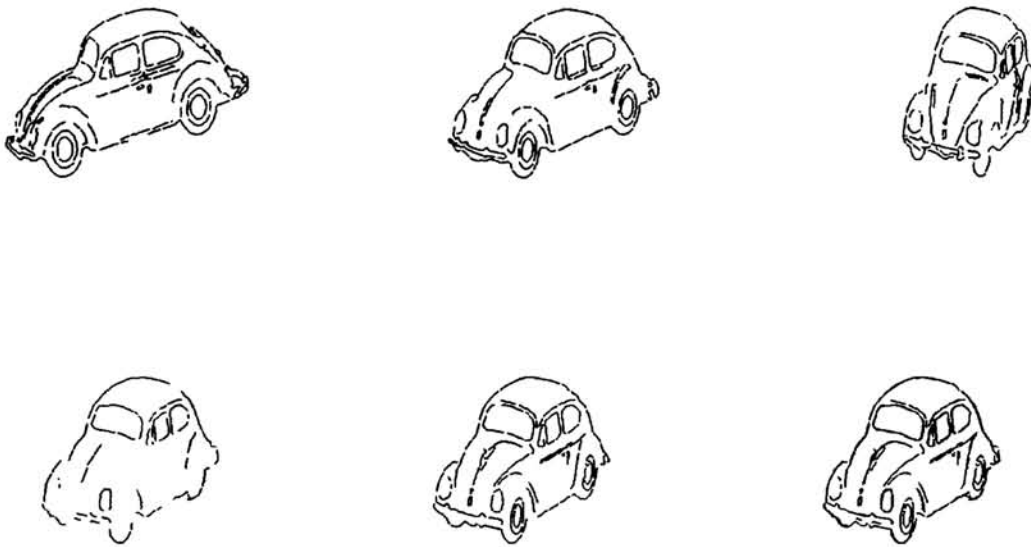

Figure 2: Top: three model pictures of a VW car. Bottom: a linear combination of the three images (left), an actual edge image (middle), and the two images overlayed (right).

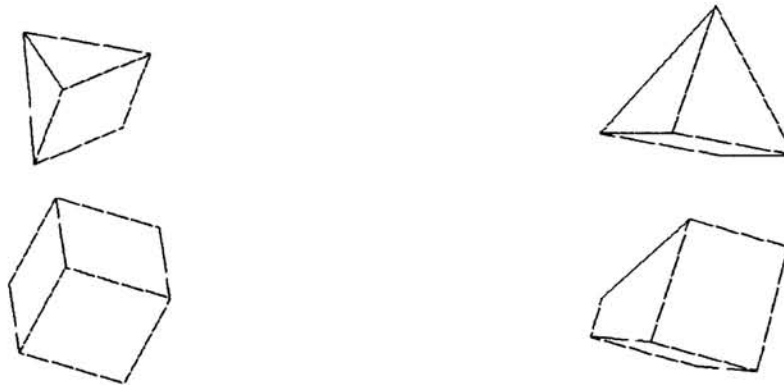

Figure 3: Top: applying an associative pyramidal operator to a pyramid (left) returns a model view of the pyramid (right, compare with Figure 1 top left). Bottom: applying the same operator to a cube (left) returns an unfamiliar image (right).

$\mathbf{z} = (z_1, ..., z_n)$ such that $(x_i, y_i, z_i) \in O$ for every $1 \le i \le n$. Let $\hat{P}$ be a view of the object, and let $\hat{\mathbf{x}} = (\hat{x}_1, ..., \hat{x}_n)$ and $\hat{\mathbf{y}} = (\hat{y}_1, ..., \hat{y}_n)$ such that $(\hat{x}_i, \hat{y}_i)$ is the position of $(x_i, y_i, z_i)$ in $\hat{P}$. We call $\mathbf{x}$, $\mathbf{y}$, and $\mathbf{z}$ the coordinate vectors of $O$, and $\hat{\mathbf{x}}$ and $\hat{\mathbf{y}}$ the corresponding coordinate vectors in $\hat{P}$. Assume $\hat{P}$ is obtained from $O$ by applying a rotation matrix $R$, a scale factor $s$, and a translation vector $(t_x, t_y)$ followed by an orthographic projection.

**Claim:**    There exist coefficients $a_1, a_2, a_3, a_4$ and $b_1, b_2, b_3, b_4$ such that:

$$\hat{\mathbf{x}} = a_1\mathbf{x} + a_2\mathbf{y} + a_3\mathbf{z} + a_4\mathbf{1}$$
$$\hat{\mathbf{y}} = b_1\mathbf{x} + b_2\mathbf{y} + b_3\mathbf{z} + b_4\mathbf{1}$$

where $\mathbf{1} = (1, ..., 1) \in \mathcal{R}^n$.

**Proof:**    Simply by assigning:

$$
\begin{aligned}
a_1 &= sr_{11} & b_1 &= sr_{21} \\
a_2 &= sr_{12} & b_2 &= sr_{22} \\
a_3 &= sr_{13} & b_3 &= sr_{23} \\
a_4 &= t_x & b_4 &= t_y
\end{aligned}
$$

Therefore, $\hat{\mathbf{x}}, \hat{\mathbf{y}} \in span\{\mathbf{x}, \mathbf{y}, \mathbf{z}, \mathbf{1}\}$ regardless of the viewpoint from which $\hat{\mathbf{x}}$ and $\hat{\mathbf{y}}$ are taken. Notice that the set of views of a rigid object does not occupy the entire linear 4-D space. Rather, the coefficients follow in addition two quadratic constraints:

$$a_1^2 + a_2^2 + a_3^2 = b_1^2 + b_2^2 + b_3^2$$
$$a_1 b_1 + a_2 b_2 + a_3 b_3 = 0$$

# Appendix B

A "recognition matrix" is defined as follows. Let $\{\mathbf{p}_1, ..., \mathbf{p}_k\}$ be a set of $k$ linearly independent vectors representing the model pictures. Let $\{\mathbf{p}_{k+1}, ..., \mathbf{p}_n\}$ be a set of vectors such that $\{\mathbf{p}_1, ..., \mathbf{p}_n\}$ are all linearly independent. We define the following matrices:

$$
\begin{aligned}
P &= (\mathbf{p}_1, ..., \mathbf{p}_k, \mathbf{p}_{k+1}, ..., \mathbf{p}_n) \\
Q &= (\mathbf{q}, ..., \mathbf{q}, \mathbf{p}_{k+1}, ..., \mathbf{p}_n)
\end{aligned}
$$

We require that:

$$LP = Q$$

Therefore:

$$L = QP^{-1}$$

Note that since $P$ is composed of $n$ linearly independent vectors, the inverse matrix $P^{-1}$ exists, therefore $L$ can always be constructed.

## Acknowledgments

We wish to thank Yael Moses for commenting on the final version of this paper. This report describes research done at the Massachusetts Institute of Technology within the Artificial Intelligence Laboratory. Support for the laboratory's artificial

intelligence research is provided in part by the Advanced Research Projects Agency of the Department of Defense under Office of Naval Research contract N00014-85-K-0124. Ronen Basri is supported by the McDonnell-Pew and the Rothchild postdoctoral fellowships.

## Footnotes

*Also, Weizmann Inst. of Science, Dept. of Applied Math., Rehovot 76100, Israel

## References

Abu-Mostafa, Y.S. & Pslatis, D. 1987. Optical neural computing. *Scientific American, 256*, 66-73.

Chien, C.H. & Aggarwal, J.K., 1987. Shape recognition from single silhouette. *Proc. of ICCV Conf. (London)* 481-490.

Faugeras, O.D. & Hebert, M., 1986. The representation, recognition and location of 3-D objects. *Int. J. Robotics Research*, 5(3), 27-52.

Fischler, M.A. & Bolles, R.C., 1981. Random sample consensus: a paradigm for model fitting with application to image analysis and automated cartography. *Communications of the ACM*, 24(6), 381-395.

Huttenlocher, D.P. & Ullman, S., 1987. Object recognition using alignment. *Proc. of ICCV Conf. (London)*, 102-111.

Koenderink, J.J. & Van Doorn, A.J., 1979. The internal representation of solid shape with respect to vision. *Biol. Cybernetics 32*, 211-216.

Kohonen, T., Oja, E., & Lehtiö, P., 1981. Storage and processing of information in distributed associative memory systems. *in Hinton, G.E. & Anderson, J.A., Parallel Models of Associative Memory. Hillsdale, NJ: Lawrence Erlbaum Associates*, 105-143.

Lowe, D.G., 1985. *Perceptual Organization and Visual Recognition.* Boston: Kluwer Academic Publishing.

Marr, D. & Ullman, S., 1981. Directional selectivity and its use in early visual processing. *Proc. R. Soc. Lond. B 211*, 151-180.

Poggio, T. & Edelman, S., 1990. A network that learns to recognize three dimensional objects. *Nature, Vol. 343*, 263-266.

Thompson, D.W. & Mundy J.L., 1987. Three dimensional model matching from an unconstrained viewpoint. *Proc. IEEE Int. Conf. on robotics and Automation*, Raleigh, N.C., 208-220.

S. Ullman and R. Basri, 1991. Recognition by Linear Combinations of Models. *IEEE Trans. on Pattern Analysis and Machine Intelligence, Vol. 13, No. 10*, pp. 992-1006

Ullman, S., 1989. Aligning pictorial descriptions: An approach to object recognition: *Cognition, 32(3)*, 193-254. Also: 1986, *A.I. Memo 931, The Artificial Intelligence Lab., M.I.T..*

Yeates, M.C., 1991. A neural network for computing the pseudo-inverse of a matrix and application to Kalman filtering. *Tech. Report, California Institute of Technology.*

Zipser, D. & Andersen, R.A., 1988. A back-propagation programmed network that simulates response properties of a subset of posterior parietal neurons. *Nature, 331*, 679-684.